# Sparse Bayesian Multi-Task Learning

**Cédric Archambeau, Shengbo Guo, Onno Zoeter**
Xerox Research Centre Europe
{Cedric.Archambeau, Shengbo.Guo, Onno.Zoeter}@xrce.xerox.com

## Abstract

We propose a new sparse Bayesian model for multi-task regression and classification. The model is able to capture correlations between tasks, or more specifically a low-rank approximation of the covariance matrix, while being sparse in the features. We introduce a general family of group sparsity inducing priors based on matrix-variate Gaussian scale mixtures. We show the amount of sparsity can be learnt from the data by combining an approximate inference approach with type II maximum likelihood estimation of the hyperparameters. Empirical evaluations on data sets from biology and vision demonstrate the applicability of the model, where on both regression and classification tasks it achieves competitive predictive performance compared to previously proposed methods.

## 1 Introduction

Learning multiple related tasks is increasingly important in modern applications, ranging from the prediction of tests scores in social sciences and the classification of protein functions in systems biology to the categorisation of scenes in computer vision and more recently to web search and ranking. In many real life problems multiple related target variables need to be predicted from a single set of input features. A problem that attracted considerable interest in recent years is to label an image with (text) keywords based on the features extracted from that image [26]. In general, this multi-label classification problem is challenging as the number of classes is equal to the vocabulary size and thus typically very large. While capturing correlations between the labels seems appealing it is in practice difficult as it rapidly leads to numerical problems when estimating the correlations.

A naive solution is to learn a model for each task separately and to make predictions using the independent models. Of course, this approach is unsatisfactory as it does not take advantage of all the information contained in the data. If the model is able to capture the task relatedness, it is expected to have generalisation capabilities that are drastically increased. This motivated the introduction of the multi-task learning paradigm that exploits the correlations amongst multiple tasks by learning them simultaneously rather than individually [12]. More recently, the abundant literature on multi-task learning demonstrated that performance indeed improves when the tasks are related [6, 31, 2, 14, 13].

The multi-task learning problem encompasses two main settings. In the first one, for every input, every task produces an output. If we restrict ourselves to multiple regression for the time being, the most basic multi-task model would consider $P$ correlated tasks[1], the vector of covariates and targets being respectively denoted by $\mathbf{x}_n \in \mathbb{R}^D$ and $\mathbf{y}_n \in \mathbb{R}^P$:

$$\mathbf{y}_n = \mathbf{W}\mathbf{x}_n + \boldsymbol{\mu} + \boldsymbol{\epsilon}_n, \qquad\qquad \boldsymbol{\epsilon}_n \sim \mathcal{N}(\mathbf{0}, \boldsymbol{\Sigma}), \qquad\qquad (1)$$

where $\mathbf{W} \in \mathbb{R}^{P \times D}$ is the matrix of weights and $\boldsymbol{\mu} \in \mathbb{R}^P$ the task offsets and $\boldsymbol{\epsilon}_n \in \mathbb{R}^P$ the vector residual errors with covariance $\boldsymbol{\Sigma} \in \mathbb{R}^{P \times P}$. In this setting, the output of *all* tasks is observed for

every input. In the second setting, the goal is to learn from a set of observed tasks and to generalise to a *new* task. This approach views the multi-task learning problem as a transfer learning problem, where it is assumed that the various tasks belong in some sense to the same environment and share common properties [23, 5]. In general only a single task output is observed for every input.

A recent trend in multi-task learning is to consider sparse solutions to facilitate the interpretation. Many formulate the sparse multi-task learning problem in a (relaxed) convex optimization framework [5, 22, 35, 23]. If the regularization constant is chosen using cross-validation, regularization-based approaches often overestimate the support [32] as they select more features than the set that generated the data. Alternatively, one can adopt a Bayesian approach to sparsity in the context of multi-task learning [29, 21]. The main advantage of the Bayesian formalism is that it enables us to learn the degree of sparsity supported by the data and does not require the user to specify the type of penalisation in advance.

In this paper, we adopt the first setting for multi-task learning, but we will consider a hierarchical Bayesian model where the entries of $\mathbf{W}$ are correlated so that the residual errors are uncorrelated. This is similar in spirit as the approach taken by [18], where tasks are related through a shared kernel matrix. We will consider a matrix-variate prior to simultaneously model task correlations and group sparsity in $\mathbf{W}$. A matrix-variate Gaussian prior was used in [35] in a maximum likelihood setting to capture task correlations and feature correlations. While we are also interested in task correlations, we will consider matrix-variate Gaussian scale mixture priors centred at zero to drive entire blocks of $\mathbf{W}$ to zero. The Bayesian group LASSO proposed in [30] is a special case. Group sparsity [34] is especially useful in presence of categorical features, which are in general represented as groups of "dummy" variables. Finally, we will allow the covariance to be of low-rank so that we can deal with problems involving a very large number of tasks.

## 2   Matrix-variate Gaussian prior

Before starting our discussion of the model, we introduce the matrix variate Gaussian as it plays a key role in our work. For a matrix $\mathbf{W} \in \mathbb{R}^{P \times D}$, the matrix-variate Gaussian density [16] with mean matrix $\mathbf{M} \in \mathbb{R}^{P \times D}$, row covariance $\mathbf{\Omega} \in \mathbb{R}^{D \times D}$ and column covariance $\mathbf{\Sigma} \in \mathbb{R}^{P \times P}$ is given by

$$\mathcal{N}(\mathbf{M}, \mathbf{\Omega}, \mathbf{\Sigma}) \propto e^{-\frac{1}{2}\mathrm{vec}(\mathbf{W}-\mathbf{M})^{\top}(\mathbf{\Omega} \otimes \mathbf{\Sigma})^{-1}\mathrm{vec}(\mathbf{W}-\mathbf{M})} \propto e^{-\frac{1}{2}\mathrm{tr}\{\mathbf{\Omega}^{-1}(\mathbf{W}-\mathbf{M})^{\top}\mathbf{\Sigma}^{-1}(\mathbf{W}-\mathbf{M})\}}. \tag{2}$$

If we let $\mathbf{\Sigma} = E(\mathbf{W} - \mathbf{M})(\mathbf{W} - \mathbf{M})^{\top}$, then $\mathbf{\Omega} = E(\mathbf{W} - \mathbf{M})^{\top}(\mathbf{W} - \mathbf{M})/c$ where $c$ ensures the density integrates to one. While this introduces a scale ambiguity between $\mathbf{\Sigma}$ and $\mathbf{\Omega}$ (easily removed by means of a prior), the use of a matrix-variate formulation is appealing as it makes explicit the structure $\mathrm{vec}(\mathbf{W})$, which is a vector formed by the concatenation of the columns of $\mathbf{W}$. This structure is reflected in its covariance matrix which is not of full rank, but is obtained by computing the Kronecker product of the row and the column covariance matrices.

It is interesting to compare a matrix-variate prior for $\mathbf{W}$ in (1) with the classical multi-level approach to multiple regression from statistics (see e.g. [20]). In a standard multi-level model, the rows of $\mathbf{W}$ are drawn iid from a multivariate Gaussian with mean $\mathbf{m}$ and covariance $\mathbf{S}$, and $\mathbf{m}$ is further drawn from zero mean Gaussian with covariance $\mathbf{R}$. Integrating out $\mathbf{m}$ leads then to a Gaussian distributed $\mathrm{vec}(\mathbf{W})$ with mean zero and with a covariance matrix that has the block diagonal elements equal to $\mathbf{S} + \mathbf{R}$ and all off-diagonal elements equal to $\mathbf{R}$. Hence, the standard multi-level model assumes a very different covariance structure than the one based on (2) and incidentally cannot learn correlated and anti-correlated tasks simultaneously.

## 3   A general family of group sparsity inducing priors

We seek a solution for which the expectation of $\mathbf{W}$ is sparse, i.e., blocks of $\mathbf{W}$ are driven to zero. A straightforward way to induce sparsity, and which would be equivalent to $\ell_1$-regularisation on blocks of $\mathbf{W}$, is to consider a Laplace prior (or double exponential). Although applicable in a penalised likelihood framework, the Laplace prior would be computationally hard in a Bayesian setting as it is not conjugate to the Gaussian likelihood. Hence, naively using this prior would prevent us from computing the posterior in closed form, even in a variational setting. In order to circumvent this problem, we take a hierarchical Bayesian approach.

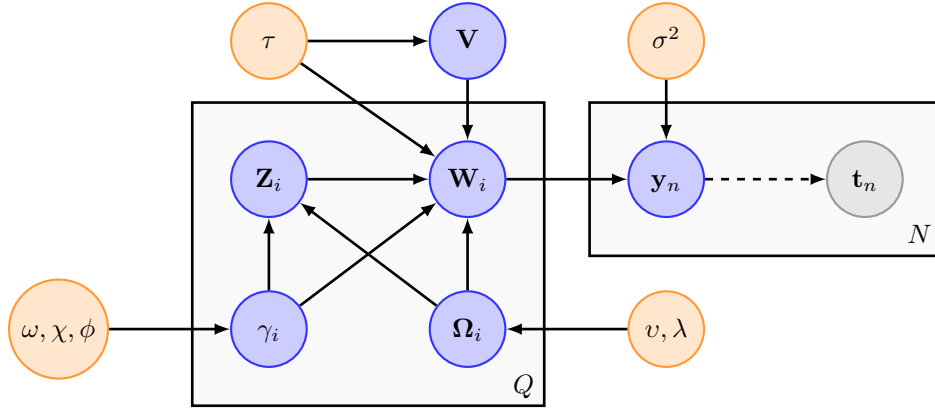

Figure 1: Graphical model for sparse Bayesian multiple regression (when excluding the dashed arrow) and sparse Bayesian multiple classification (when considering all arrows).

We assume that the marginal prior, or *effective prior*, on each block $\mathbf{W}_i \in \mathbb{R}^{P \times D_i}$ has the form of a matrix-variate Gaussian scale mixture, a generalisation of the multivariate Gaussian scale mixture [3]:

$$p(\mathbf{W}_i) = \int_0^\infty \mathcal{N}(\mathbf{0}, \gamma_i^{-1} \mathbf{\Omega}_i, \mathbf{\Sigma}) \, p(\gamma_i) \, d\gamma_i, \qquad \sum_{i=1}^Q D_i = D, \qquad (3)$$

where $\mathbf{\Omega}_i \in \mathbb{R}^{D_i \times D_i}$, $\mathbf{\Sigma} \in \mathbb{R}^{P \times P}$ and $\gamma_i > 0$ is the latent precision (i.e., inverse scale) associated to block $\mathbf{W}_i$.

A sparsity inducing prior for $\mathbf{W}_i$ can then be constructed by choosing a suitable hyperprior for $\gamma_i$. We impose a generalised inverse Gaussian prior (see Supplemental Appendix A for a formal definition with special cases) on the latent precision variables:

$$\gamma_i \sim \mathcal{N}^{-1}(\omega, \chi, \phi) = \frac{\chi^{-\omega} \left(\sqrt{\chi\phi}\right)^\omega}{2 K_\omega(\sqrt{\chi\phi})} \gamma_i^{\omega-1} e^{-\frac{1}{2}(\chi\gamma_i^{-1} + \phi\gamma_i)}, \qquad (4)$$

where $K_\omega(\cdot)$ is the modified Bessel function of the second kind, $\omega$ is the index, $\sqrt{\chi\phi}$ defines the concentration of the distribution and $\sqrt{\chi/\phi}$ defines its scale. The effective prior is then a symmetric matrix-variate generalised hyperbolic distribution:

$$p(\mathbf{W}_i) \propto \frac{K_{\omega + \frac{PD_i}{2}}\left(\sqrt{\chi(\phi + \mathrm{tr}\{\mathbf{\Omega}_i^{-1}\mathbf{W}_i^\top\mathbf{\Sigma}^{-1}\mathbf{W}_i\})}\right)}{\left(\sqrt{\frac{\phi + \mathrm{tr}\{\mathbf{\Omega}_i^{-1}\mathbf{W}_i^\top\mathbf{\Sigma}^{-1}\mathbf{W}_i\}}{\chi}}\right)^{\omega + \frac{PD_i}{2}}}. \qquad (5)$$

The marginal (5) has fat tails compared to the matrix-variate Gaussian. In particular, the family contains the matrix-variate Student-$t$, the matrix-variate Laplace and the matrix-variate Variance-Gamma as special cases. Several of the multivariate equivalents have recently been used as priors to induce sparsity in the Bayesian paradigm, both in the context of supervised [19, 11] and unsupervised linear Gaussian models [4].

## 4 Sparse Bayesian multiple regression

We view $\{\mathbf{W}_i\}_{i=1}^Q$, $\{\mathbf{\Omega}_i\}_{i=1}^Q$ and $\{\gamma_1, \ldots, \gamma_{D_1}, \ldots, \gamma_1, \ldots, \gamma_{D_Q}\}$ as latent variables that need to be marginalised over. This is motivated by the fact that overfitting is avoided by integrating out all parameters whose cardinality scales with the model complexity, i.e., the number of dimensions and/or the number of tasks. We further introduce a latent projectoin matrix $\mathbf{V} \in \mathbb{R}^{P \times K}$ and a set of latent matrices $\{\mathbf{Z}_i\}_{i=1}^Q$ to make a low-rank approximation of the column covariance $\mathbf{\Sigma}$ as explained below. Note also that $\mathbf{\Omega}_i$ captures the correlations between the rows of group $i$.

The complete probabilistic model is given by

$$\mathbf{y}_n|\mathbf{W},\mathbf{x}_n \sim \mathcal{N}(\mathbf{W}\mathbf{x}_n,\sigma^2\mathbf{I}_P), \qquad\qquad \mathbf{V} \sim \mathcal{N}(\mathbf{0},\tau\mathbf{I}_P,\mathbf{I}_K), \qquad (6)$$

$$\mathbf{W}_i|\mathbf{V},\mathbf{Z}_i,\mathbf{\Omega}_i,\gamma_i \sim \mathcal{N}(\mathbf{V}\mathbf{Z}_i,\gamma_i^{-1}\mathbf{\Omega}_i,\tau\mathbf{I}_P), \qquad\qquad \mathbf{\Omega}_i \sim \mathcal{W}^{-1}(v,\lambda\mathbf{I}_{D_i}),$$

$$\mathbf{Z}_i|\mathbf{\Omega}_i,\gamma_i \sim \mathcal{N}(\mathbf{0},\gamma_i^{-1}\mathbf{\Omega}_i,\mathbf{I}_K), \qquad\qquad \gamma_i \sim \mathcal{N}^{-1}(\omega,\chi,\phi),$$

where $\sigma^2$ is the residual noise variance and $\tau$ is residual variance associated to $\mathbf{W}$. The graphical model is shown in Fig. 1. We reparametrise the inverse Wishart distribution and define it as follows:

$$\mathbf{\Omega} \sim \mathcal{W}^{-1}(v,\mathbf{\Lambda}) = \frac{|\mathbf{\Lambda}|^{\frac{D+v-1}{2}}|\mathbf{\Omega}^{-1}|^{\frac{2D+v}{2}}}{2^{\frac{(D+v-1)D}{2}}\Gamma_D\left(\frac{D+v-1}{2}\right)}e^{-\frac{1}{2}\text{tr}\{\mathbf{\Lambda}\mathbf{\Omega}^{-1}\}}, \quad v > 0,$$

where $\Gamma_p(z) = \pi^{\frac{p(p-1)}{4}}\prod_{j=1}^p \Gamma(z+\frac{1-j}{2})$.

Using the compact notations $\mathbf{W} = (\mathbf{W}_1,\ldots,\mathbf{W}_Q)$, $\mathbf{Z} = (\mathbf{Z}_1,\ldots,\mathbf{Z}_Q)$, $\mathbf{\Omega} = \text{diag}\{\mathbf{\Omega}_1,\ldots,\mathbf{\Omega}_Q\}$ and $\mathbf{\Gamma} = \text{diag}\{\gamma_1,\ldots,\gamma_{D_1},\ldots,\gamma_1,\ldots,\gamma_{D_Q}\}$, we can compute the following marginal:

$$p(\mathbf{W}|\mathbf{V},\mathbf{\Omega}) \propto \iint \mathcal{N}(\mathbf{V}\mathbf{Z},\mathbf{\Gamma}^{-1}\mathbf{\Omega},\tau\mathbf{I}_P)\mathcal{N}(\mathbf{0},\mathbf{\Gamma}^{-1}\mathbf{\Omega},\mathbf{I}_K)p(\mathbf{\Gamma})d\mathbf{Z}d\mathbf{\Gamma}$$

$$= \int \mathcal{N}(\mathbf{0},\mathbf{\Gamma}^{-1}\mathbf{\Omega},\mathbf{V}\mathbf{V}^\top + \tau\mathbf{I}_P)p(\mathbf{\Gamma})d\mathbf{\Gamma}.$$

Thus, the probabilistic model induces sparsity in the blocks of $\mathbf{W}$, while taking correlations between the task parameters into account through the random matrix $\mathbf{\Sigma} \approx \mathbf{V}\mathbf{V}^\top + \tau\mathbf{I}_P$. This is especially useful when there is a very large number of tasks.

The latent variables $\mathcal{Z} = \{\mathbf{W},\mathbf{V},\mathbf{Z},\mathbf{\Omega},\mathbf{\Gamma}\}$ are infered by variational EM [27], while the hyperparameters $\vartheta = \{\sigma^2,\tau,\upsilon,\lambda,\omega,\chi,\phi\}$ are estimated by type II ML [8, 25]). Using variational inference is motivated by the fact that deterministic approximate inference schemes converge faster than traditional sampling methods such as Markov chain Monte Carlo (MCMC), and their convergence can easily be monitored. The choice of learning the hyperparameters by type II ML is preferred to the option of placing vague priors over them, although this would also be a valid option.

In order to find a tractable solution, we assume that the variational posterior $q(\mathcal{Z}) = q(\mathbf{W},\mathbf{V},\mathbf{Z},\mathbf{\Omega},\mathbf{\Gamma})$ factorises as $q(\mathbf{W})q(\mathbf{V})q(,\mathbf{Z})q(\mathbf{\Omega})q(\mathbf{\Gamma})$ given the data $\mathcal{D} = \{(\mathbf{y}_n,\mathbf{x}_n)\}_{n=1}^N$ [7]. The variational EM combined to the type II ML estimation of the hyperparameters cycles through the following two steps until convergence:

1. Update of the approximate posterior of the latent variables and parameters for fixed hyperparameters. The update for $\mathbf{W}$ is given by

$$q(\mathbf{W}) \propto e^{\langle\ln p(\mathcal{D},\mathcal{Z}|\vartheta)\rangle_{q(\mathcal{Z}/\mathbf{W})}}, \qquad (7)$$

where $\mathcal{Z}/\mathbf{W}$ is the set $\mathcal{Z}$ with $\mathbf{W}$ removed and $\langle\cdot\rangle_q$ denotes the expectation with respect to $q$. The posteriors of the other latent matrices have the same form.

2. Update of the hyperparameters for fixed variational posteriors:

$$\vartheta \leftarrow \underset{\vartheta}{\text{argmax}}\, \langle\ln p(\mathcal{D},\mathcal{Z},|\vartheta)\rangle_{q(\mathcal{Z})}. \qquad (8)$$

Variational EM converges to a local maximum of the log-marginal likelihood. The convergence can be checked by monitoring the variational lower bound, which monotonically increases during the optimisation. Next, we give the explicit expression of the variational EM steps and the updates for the hyperparameters, whereas we show that of the variational bound in the Supplemental Appendix D.

### 4.1 Variational E step (mean field)

Asssuming a factorised posterior enables us to compute it in closed form as the priors are each conjugate to the Gaussian likelihood. The approximate posterior is given by

$$q(\mathcal{Z}) = \mathcal{N}(\mathbf{M}_W,\mathbf{\Omega}_W,\mathbf{S}_W)\mathcal{N}(\mathbf{M}_V,\mathbf{\Omega}_V,\mathbf{S}_V)\mathcal{N}(\mathbf{M}_Z,\mathbf{\Omega}_Z,\mathbf{S}_Z) \qquad (9)$$

$$\times \prod_i \mathcal{W}^{-1}(\upsilon_i,\mathbf{\Lambda}_i)\mathcal{N}^{-1}(\omega_i,\chi_i,\phi_i).$$

The expression of posterior parameters are given in Supplemental Appendix C. The computational bottleneck resides in the inversion of $\mathbf{\Omega}_W$ which is $\mathcal{O}(D^3)$ per iteration. When $D > N$, we can use the Woodbury identity for a matrix inversion of complexity $\mathcal{O}(N^3)$ per iteration.

## 4.2 Hyperparameter updates

To learn the degree of sparsity from data we optimise the hyperparameters. There are no closed form updates for $\{\omega, \chi, \phi\}$. Hence, we need to find the root of the following expressions, e.g., by line search:

$$\omega: \quad Q \ln \sqrt{\frac{\phi}{\chi}} - Q \frac{d \ln K_\omega(\sqrt{\chi \phi})}{d\omega} \sum_i \langle \ln \gamma_i \rangle = 0, \tag{10}$$

$$\chi: \quad \frac{Q\omega}{\chi} - \frac{Q}{2} \sqrt{\frac{\phi}{\chi}} R_\omega(\sqrt{\chi \phi}) + \frac{1}{2} \sum_i \langle \gamma_i^{-1} \rangle = 0, \tag{11}$$

$$\phi: \quad Q \sqrt{\frac{\chi}{\phi}} R_\omega(\sqrt{\chi \phi}) - \sum_i \langle \gamma_i \rangle = 0, \tag{12}$$

where (**??**) was invoked. Unfortunately, the derivative in the first equation needs to be estimated numerically. When considering special cases of the mixing density such as the Gamma or the inverse Gamma simplified updates are obtained and no numerical differentiation is required.

Due to space constraints, we omit the type II ML updates for the other hyperparameters.

## 4.3 Predictions

Predictions are performed by Bayesian averaging. The predictive distribution is approximated as follows: $p(\mathbf{y}_* | \mathbf{x}_*) \approx \int p(\mathbf{y}_* | \mathbf{W}, \mathbf{x}_*) q(\mathbf{W}) d\mathbf{W} = \mathcal{N}(\mathbf{M}_W \mathbf{x}_*, (\sigma^2 + \mathbf{x}_*^\top \mathbf{\Omega}_W \mathbf{x}_*) \mathbf{I}_P)$.

# 5 Sparse Bayesian multiple classification

We restrict ourselves to multiple binary classifiers and consider a probit model in which the likelihood is derived from the Gaussian cumulative density. A probit model is equivalent to a Gaussian noise and a step function likelihood [1]. Let $\mathbf{t}_n \in \mathbb{R}^P$ be the class label vectors, with $t_{np} \in \{-1, +1\}$ for all $n$. The likelihood is replaced by

$$\mathbf{t}_n | \mathbf{y}_n \sim \prod_p I(t_{np} y_{np}), \qquad\qquad \mathbf{y}_n | \mathbf{W}, \mathbf{x}_n \sim \mathcal{N}(\mathbf{W} \mathbf{x}_n, \sigma^2 \mathbf{I}_P), \tag{13}$$

where $I(z) = 1$ for $z \geqslant 0$ and $0$ otherwise. The rest of the model is as before; we will set $\sigma = 1$.

The latent variables to infer are now $\mathbf{Y}$ and $\mathbf{Z}$. Again, we assume a factorised posterior. We further assume the variational posterior $q(\mathbf{Y})$ is a product of truncated Gaussians (see Supplemental Appendix B):

$$q(\mathbf{Y}) \propto \prod_n \prod_p I(t_{np} y_{np}) \mathcal{N}(\nu_{np}, 1) = \prod_{t_{np}=+1} \mathcal{N}_+(\nu_{np}, 1) \prod_{t_{np}=-1} \mathcal{N}_-(\nu_{np}, 1), \tag{14}$$

where $\nu_{np}$ is the $p^{\text{th}}$ entry of $\boldsymbol{\nu}_n = \mathbf{M}_W \mathbf{x}_n$. The other variational and hyperparameter updates are unchanged, except that $\mathbf{Y}$ is replaced by matrix $\boldsymbol{\nu}_\pm$. The elements of $\boldsymbol{\nu}_\pm$ are defined in (**??**).

## 5.1 Bayesian classification

In Bayesian classification the goal is to predict the label with highest posterior probability. Based on the variational approximation we propose the following classification rule:

$$\hat{\mathbf{t}}_* = \arg\max_{\mathbf{t}_*} P(\mathbf{t}_* | \mathbf{T}) \approx \arg\max_{\mathbf{t}_*} \prod_p \int \mathcal{N}_{t_{*p}}(\nu_{*p}, 1) dy_{*p} = \arg\max_{\mathbf{t}_*} \prod_p \Phi\left(t_{*p} \nu_{*p}\right), \tag{15}$$

where $\boldsymbol{\nu}_* = \mathbf{M}_W \mathbf{x}_*$. Hence, to decide whether the label $t_{*p}$ is $-1$ or $+1$ it is sufficient to use the sign of $\nu_{*p}$ as the decision rule. However, the probability $P(t_{*p} | \mathbf{T})$ tells us also how confident we are in the prediction we make.

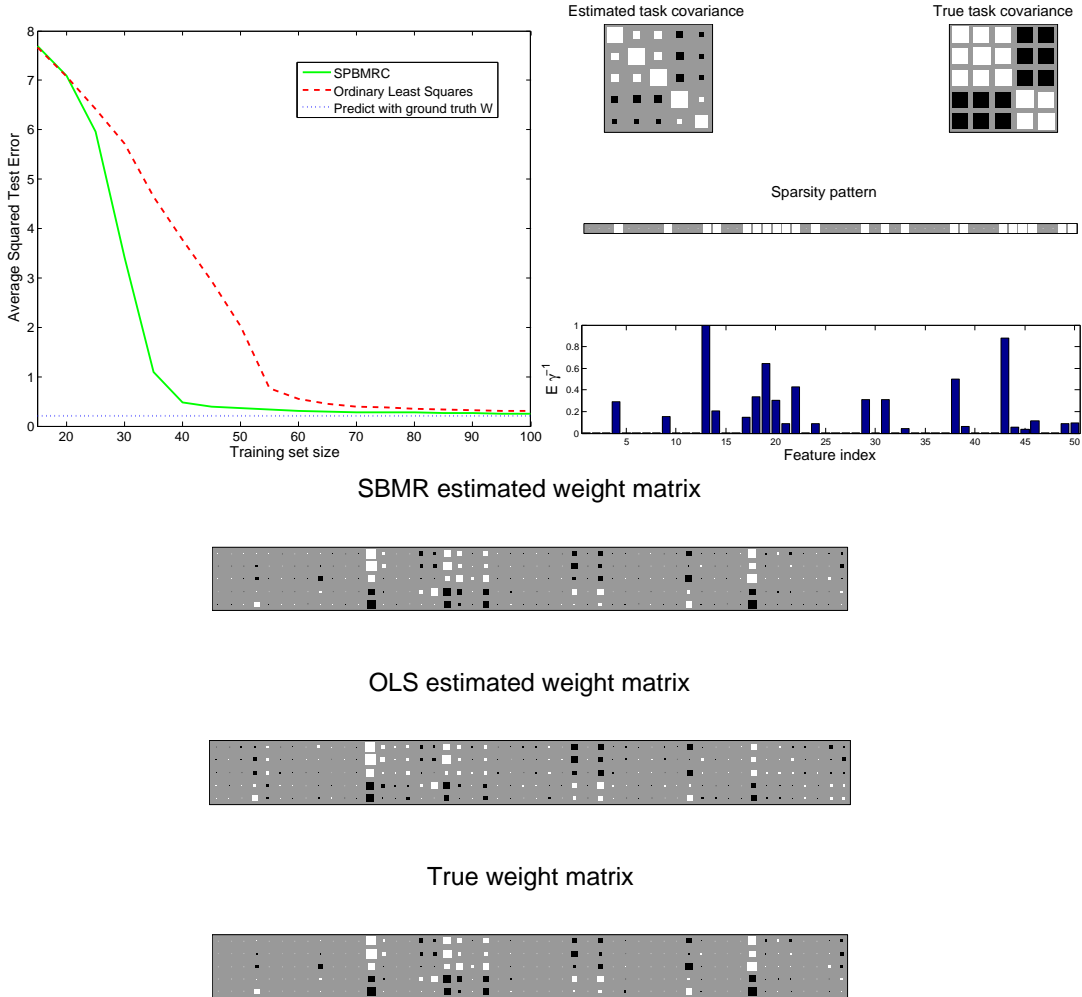

Figure 2: Results for the ground truth data set. Top left: Prediction accuracy on a test set as a function of training set size. Top right: estimated and true $\mathbf{\Sigma}$ (top), true underlying sparsity pattern (middle) and inverse of the posterior mean of $\{\gamma_i\}_i$ showing that the sparsity is correctly captured (bottom). Bottom diagrams: Hinton diagram of true $\mathbf{W}$ (bottom), ordinary least squares learnt $\mathbf{W}$ (middle) and the sparse Bayesian multi-task learnt $\mathbf{W}$ (top). The ordinary least squares learnt $\mathbf{W}$ contains many non-zero elements.

# 6 A model study with ground truth data

To understand the properties of the model we study a regression problem with known parameters. Figure 2 shows the results for 5 tasks and 50 features. Matrix $\mathbf{W}$ is drawn using $\mathbf{V} = [\sqrt{.9} \ \sqrt{.9} \ \sqrt{.9} \ -\sqrt{.9} \ -\sqrt{.9}]^\top$ and $\tau = 0.1$, i.e. the covariance for $\mathrm{vec}(\mathbf{W})$ has 1's on the diagonal and $\pm.9$ on the off-diagonal elements. The first three tasks and the last two tasks are positively correlated. There is a negative correlation between the two groups. The active features are randomly selected among the 50 candidate features. We evaluate the models with $10^4$ test points and repeated the experiment 25 times. Gaussian noise was added to the targets ($\sigma = 0.1$).

It can be observed that the proposed model performs better and converges faster to the optimal performance when the data set size increases compared ordinary least squares. Note also that both $\mathbf{\Sigma}$ and the sparsity pattern are correctly identified.

Table 1: Performance (with standard deviation) of classification tasks on Yeast and Scene data sets in terms of accuracy and AUC. LR: Bayesian logistic regression; Pooling: pooling all data and learning a single model; Xue: the matrix stick-breaking process based multi-task learning model proposed in [33]. $K = 10$ for the proposed models (i.e., Laplace, Student-t, and ARD). Note that the first five rows for Yeast and Scene data sets are reported in [29]. The reported performances are averaged over five randomized repetitions.

| Model | Yeast | | Scene | |
|---|---|---|---|---|
| | Accuracy | AUC | Accuracy | AUC |
| LR | 0.5047 | 0.5049 | 0.7362 | 0.6153 |
| Pool | 0.4983 | 0.5112 | 0.7862 | 0.5433 |
| Xue [33] | 0.5106 | 0.5105 | 0.7765 | 0.5603 |
| Model-1 [29] | 0.5212 | 0.5244 | 0.7756 | 0.6325 |
| Model-2 [29] | 0.5424 | 0.5406 | 0.7911 | 0.6416 |
| Chen [15] | NA | 0.7987±0.0044 | NA | 0.9160±0.0038 |
| Laplace | 0.7987±0.0017 | 0.8349±0.0020 | 0.8892±0.0038 | 0.9188±0.0041 |
| Student | 0.7988±0.0017 | 0.8349±0.0019 | 0.8897±0.0034 | 0.9183±0.0041 |
| ARD | 0.7987±0.0020 | 0.8349±0.0020 | 0.8896±0.0044 | 0.9187±0.0042 |

## 7  Multi-task classification experiments

In this section, we evaluate the proposed model on two data sets: Yeast [17] and Scene [9], which have been widely used as testbeds to evaluate multi-task learning approaches [28, 29, 15]. To demonstrate the superiority of the proposed models, we conduct systematic empirical evaluations including the comparisons with (1) Bayesian logistic regression (BLR) that learns tasks separately, (2) a pooling model that pools data together and learns a single model collectively, and (3) the state-of-the-art multi-task learning methods proposed in [33, 29, 15].

We follow the experimental setting as introduced in [29] for fair comparisons, and omit the detailed setting due to space limitation. We evaluate all methods for the classification task using two metrics: (1) overall accuracy at a threshold of zero and (2) the average area under the curve (AUC). Results on the Yeast and Scence data sets using these two metrics are reported in Table 7. It is interesting to note that even for small values of $K$ (fewer parameters in the column covariance) the proposed model achieves good results. We also study how the performances vary with different $K$ on a tuning set, and observe that there are no significant differences on performances using different $K$ (not shown in the paper). The results in Table 7 were produced with $K = 10$.

The proposed models (Laplace, Student-t, ARD) significantly outperform the Bayesian logistic regression approach that learns each task separately. This observation agrees with the previous work [6, 31, 2, 5] demonstrating that the multi-task approach is beneficial over the naive approach of learning tasks separately. For the Yeast data set, the proposed models are significantly better than "Xue" [33], Model-1 and Model-2 [29], and the best performing model in [15]. For the Scene data set, our models and the model in [15] show comparable results.

The advantage of using hierarchical priors is particularly evident in a low data regime. To study the impact of training set size on performance, we report the accuracy and AUC as functions of the training set sizes in Figure 3. For this experiment, we use a single test set of size 1196, which replicates the experimental setup in [29]. Figure 3 shows that the proposed Bayesian methods perform well overall, but that the performances are not significantly impacted when the number of data is small. Similar results were obtained for the Yeast data set.

## 8  Conclusion

In this work we proposed a Bayesian multi-task learning model able to capture correlations between tasks and to learn the sparsity pattern of the data features simultaneously. We further proposed a low-rank approximation of the covariance to handle a very large number of tasks. Combining low-rank and sparsity at the same time has been a long open standing issue in machine learning. Here, we are able to achieve this goal by exploiting the special structure of the parameters set. Hence, the

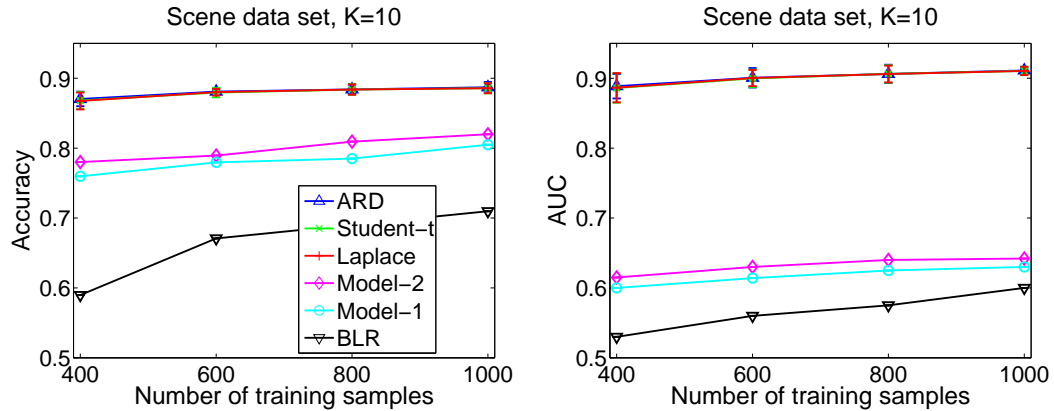

Figure 3: Model comparisons in terms of classification accuracy and AUC on the Scene data set for $K = 10$. Error bars represent 3 times the standard deviation. Results for Bayesian logistic regression (BLR), Model-1 and Model-2 are obtained based on the measurements using a ruler from Figure 2 in [29], for which no error bars are given.

proposed model combines sparsity and low-rank in a different manner than in [10], where a sum of a sparse and low-rank matrix is considered.

By considering a matrix-variate Gaussian scale mixture prior we extended the Bayesian group LASSO to a more general family of group sparsity inducing priors. This suggests the extension of current Bayesian methodology to learn structured sparsity from data in the future. A possible extension is to consider the graphical LASSO to learn sparse precision matrices $\Omega^{-1}$ abd $\Sigma^{-1}$. A similiar approach was explored in [35].

## Footnotes

[1]While it is straightforward to show that the maximum likelihood estimate of $\mathbf{W}$ would be the same as when considering uncorrelated noise, imposing any prior on $\mathbf{W}$ would lead to a different solution.

# References

[1] J. H. Albers and S. Chib. Bayesian analysis of binary and polychotomous response data. *J.A.S.A.*, 88(422):669–679, 1993.

[2] R. K. Ando and T. Zhang. A framework for learning predictive structures from multiple tasks and unlabeled data. *JMLR*, 6:1817–1853, 2005.

[3] D. F. Andrews and C. L. Mallows. Scale mixtures of normal distributions. *Journal of the Royal Statistical Society B*, 36(1):99–102, 1974.

[4] C. Archambeau and F. Bach. Sparse probabilistic projections. In *NIPS*. MIT Press, 2008.

[5] A. Argyriou, T. Evgeniou, and M. Pontil. Convex multi-task feature learning. *Machine Learning*, 73:243–272, 2008.

[6] B. Bakker and T. Heskes. Task clustering and gating for bayesian multitask learning. *JMLR*, 4:83–99, 2003.

[7] M. J. Beal. *Variational Algorithms for Approximate Bayesian Inference*. PhD thesis, Gatsby Computational Neuroscience Unit, University College London, 2003.

[8] J. O. Berger. *Statistical Decision Theory and Bayesian Analysis*. Springer, New York, 1985.

[9] M. R. Boutell, J. Luo, X. Shen, and C. M. Brown. Learning multi-label scene classification. *Pattern Recognition*, 37(9):1757–1771, 2004.

[10] E. J. Candès, X. Li, Y. Ma, and J. Wright. Robust principal component analysis? *Journal of the ACM*, 58:1–37, June 2011.

[11] F. Caron and A. Doucet. Sparse Bayesian nonparametric regression. In *ICML*, pages 88–95. ACM, 2008.

[12] R. Caruana. Multitask learning. *Machine Learning*, 28(1):41–75, 1997.

[13] O. Chapelle, P. Shivaswamy, S. Vadrevu, K. Weinberger, Y. Zhang, and B. Tseng. Multi-task learning for boosting with application to web search ranking. In *SIGKDD*, pages 1189–1198, 2010.

[14] R. Chari, W. W. Lockwood, B. P. Coe, A. Chu, D. Macey, A. Thomson, J. J. Davies, C. MacAulay, and W. L. Lam. Sigma: A system for integrative genomic microarray analysis of cancer genomes. *BMC Genomics*, 7:324, 2006.

[15] J. Chen, J. Liu, and J. Ye. Learning incoherent sparse and low-rank patterns from multiple tasks. In *SIGKDD*, pages 1179–1188. ACM, 2010.

[16] A. P. Dawid. Some matrix-variate distribution theory: Notational considerations and a bayesian application. *Biometrika*, 68(1):265–274, 1981.

[17] A. Elisseeff and J. Weston. A kernel method for multi-labelled classification. In *NIPS*. 2002.

[18] T. Evgeniou, C. A. Micchelli, and M. Pontil. Learning multiple tasks with kernel methods. *JMLR*, 6:615–637, 2005.

[19] M. Figueiredo. Adaptive sparseness for supervised learning. *IEEE Transactions on PAMI*, 25:1150–1159, 2003.

[20] A. Gelman and J. Hill. *Data Analysis Using Regression and Multilevel/Hiererarchical Models*. Cambridge University Press, 2007.

[21] D. Hernández-Lobato, J. M. Hernández-Lobato, T. Helleputte, and P. Dupont. Expectation propagation for Bayesian multi-task feature selection. In *ECML-PKDD*, pages 522–537, 2010.

[22] L. Jacob, F. Bach, and J.-P. Vert. Clustered multi-task learning: A convex formulation. In *NIPS*, pages 745–752. 2009.

[23] T. Jebara. Multitask sparsity via maximum entropy discrimination. *JMLR*, 12:75–110, 2011.

[24] B. Jørgensen. *Statistical Properties of the Generalized Inverse Gaussian Distribution*. Springer, 1982.

[25] D. J. C. MacKay. Bayesian interpolation. *Neural Computation*, 4(3):415–447, 1992.

[26] A. Makadia, V. Pavlovic, and S. Kumar. A new baseline for image annotation. In *ECCV*, 2008.

[27] R. M. Neal and G. E. Hinton. A view of the EM algorithm that justifies incremental, sparse, and other variants. In M. I. Jordan, editor, *Learning in Graphical Models*, pages 355–368. MIT press, 1998.

[28] P. Rai and H. Daume. Multi-label prediction via sparse infinite cca. In *NIPS*, pages 1518–1526. 2009.

[29] P. Rai and H. D. III. Infinite predictor subspace models for multitask learning. In *AISTATS*, pages 613–620, 2010.

[30] S. Raman, T. J. Fuchs, P. J. Wild, E. Dahl, and V. Roth. The Bayesian group-Lasso for analyzing contingency tables. In *ICML*, pages 881–888, 2009.

[31] A. Torralba, K. P. Murphy, and W. T. Freeman. Sharing features: efficient boosting procedures for multiclass object detection. In *CVPR*, pages 762–769. IEEE Computer Society, 2004.

[32] M. Wainwright. Sharp thresholds for high-dimensional and noisy sparsity recovery using $l_1$-constrained quadratic programming (lasso). *IEEE Transactions on Information Theory*, 55(5):2183 –2202, 2009.

[33] Y. Xue, D. Dunson, and L. Carin. The matrix stick-breaking process for flexible multi-task learning. In *ICML*, pages 1063–1070, 2007.

[34] M. Yuan and Y. Lin. Model selection and estimation in regression with grouped variables. *J. R. Statistic. Soc. B*, 68(1):49–67, 2006.

[35] Y. Zhang and J. Schneider. Learning multiple tasks with a sparse matrix-normal penalty. In *NIPS*, pages 2550–2558. 2010.

